# Packet Routing in Dynamically Changing Networks: A Reinforcement Learning Approach

**Justin A. Boyan**
School of Computer Science
Carnegie Mellon University
Pittsburgh, PA 15213

**Michael L. Littman***
Cognitive Science Research Group
Bellcore
Morristown, NJ 07962

## Abstract

This paper describes the Q-routing algorithm for packet routing, in which a reinforcement learning module is embedded into each node of a switching network. Only local communication is used by each node to keep accurate statistics on which routing decisions lead to minimal delivery times. In simple experiments involving a 36-node, irregularly connected network, Q-routing proves superior to a nonadaptive algorithm based on precomputed shortest paths and is able to route efficiently even when critical aspects of the simulation, such as the network load, are allowed to vary dynamically. The paper concludes with a discussion of the tradeoff between discovering shortcuts and maintaining stable policies.

## 1  INTRODUCTION

The field of reinforcement learning has grown dramatically over the past several years, but with the exception of backgammon [8, 2], has had few successful applications to large-scale, practical tasks. This paper demonstrates that the practical task of routing packets through a communication network is a natural application for reinforcement learning algorithms.

Our "Q-routing" algorithm, related to certain distributed packet routing algorithms [6, 7], learns a routing policy which balances minimizing the number of "hops" a packet will take with the possibility of congestion along popular routes. It does this by experimenting with different routing policies and gathering statistics about which decisions minimize total delivery time. The learning is continual and online, uses only local information, and is robust in the face of irregular and dynamically changing network connection patterns and load.

The experiments in this paper were carried out using a discrete event simulator to model the transmission of packets through a local area network and are described in detail in [5].

## 2   ROUTING AS A REINFORCEMENT LEARNING TASK

A packet routing policy answers the question: to which adjacent node should the current node send its packet to get it as quickly as possible to its eventual destination? Since the policy's performance is measured by the total time taken to deliver a packet, there is no "training signal" for directly evaluating or improving the policy until a packet finally reaches its destination. However, using reinforcement learning, the policy can be updated more quickly and using only local information.

Let $Q_x(d, y)$ be the time that a node $x$ estimates it takes to deliver a packet $P$ bound for node $d$ by way of $x$'s neighbor node $y$, including any time that $P$ would have to spend in node $x$'s queue.[1] Upon sending $P$ to $y$, $x$ immediately gets back $y$'s estimate for the time remaining in the trip, namely

$$t = \min_{z \in \text{neighbors of } y} Q_y(d, z)$$

If the packet spent $q$ units of time in $x$'s queue and $s$ units of time in transmission between nodes $x$ and $y$, then $x$ can revise its estimate as follows:

$$\Delta Q_x(d, y) = \eta \,(\; \overbrace{q + s + t}^{\text{new estimate}} \; - \; \overbrace{Q_x(d, y)}^{\text{old estimate}} \;)$$

where $\eta$ is a "learning rate" parameter (usually 0.5 in our experiments). The resulting algorithm can be characterized as a version of the Bellman-Ford shortest paths algorithm [1, 3] that (1) performs its path relaxation steps asynchronously and online; and (2) measures path length not merely by number of hops but rather by total delivery time.

We call our algorithm "Q-routing" and represent the Q-function $Q_x(d, y)$ by a large table. We also tried approximating $Q_x$ with a neural network (as in e.g. [8, 4]), which allowed the learner to incorporate diverse parameters of the system, including local queue size and time of day, into its distance estimates. However, the results of these experiments were inconclusive.

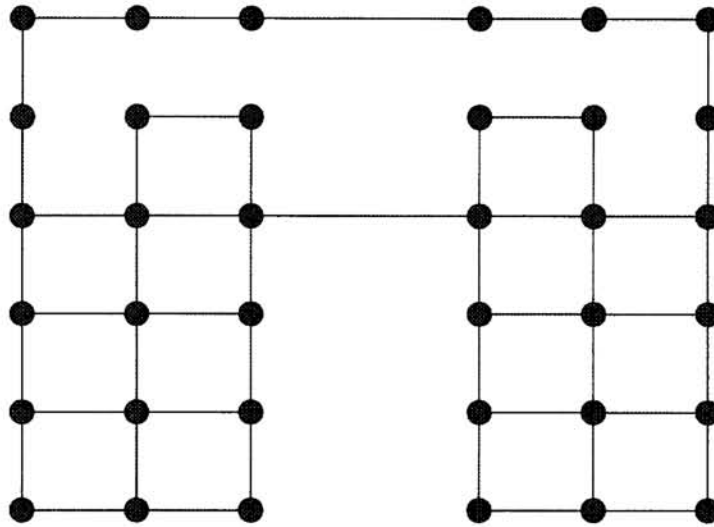

Figure 1: The irregular 6 × 6 grid topology

## 3   RESULTS

We tested the Q-routing algorithm on a variety of network topologies, including the 7-hypercube, a 116-node LATA telephone network, and an irregular 6 × 6 grid. Varying the network load, we measured the average delivery time for packets in the system after learning had settled on a routing policy, and compared these delivery times with those given by a static routing scheme based on shortest paths. The result was that in all cases, Q-routing is able to sustain a higher level of network load than could shortest paths.

This section presents detailed results for the irregular grid network pictured in Figure 1. Under conditions of low load, the network learns fairly quickly to route packets along shortest paths to their destinations. The performance vs. time curve plotted in the left part of Figure 2 demonstrates that the Q-routing algorithm, after an initial period of inefficiency during which it learns the network topology, performs about as well as the shortest path router, which is optimal under low load.

As network load increases, however, the shortest path routing scheme ceases to be optimal: it ignores the rising levels of congestion and soon floods the network with packets. The right part of Figure 2 plots performance vs. time for the two routing schemes under high load conditions: while shortest path is unable to tolerate the packet load, Q-routing learns an efficient routing policy. The reason for the learning algorithm's success is apparent in the "policy summary diagrams" in Figure 3. These diagrams indicate, for each node under a given policy, how many of the 36 × 35 point-to-point routes go through that node. In the left part of Figure 3, which summarizes the shortest path routing policy, two nodes in the center of the network (labeled 570 and 573) are on many shortest paths and thus become congested when network load is high. By contrast, the diagram on the right shows that Q-routing, under conditions of high load, has learned a policy which routes

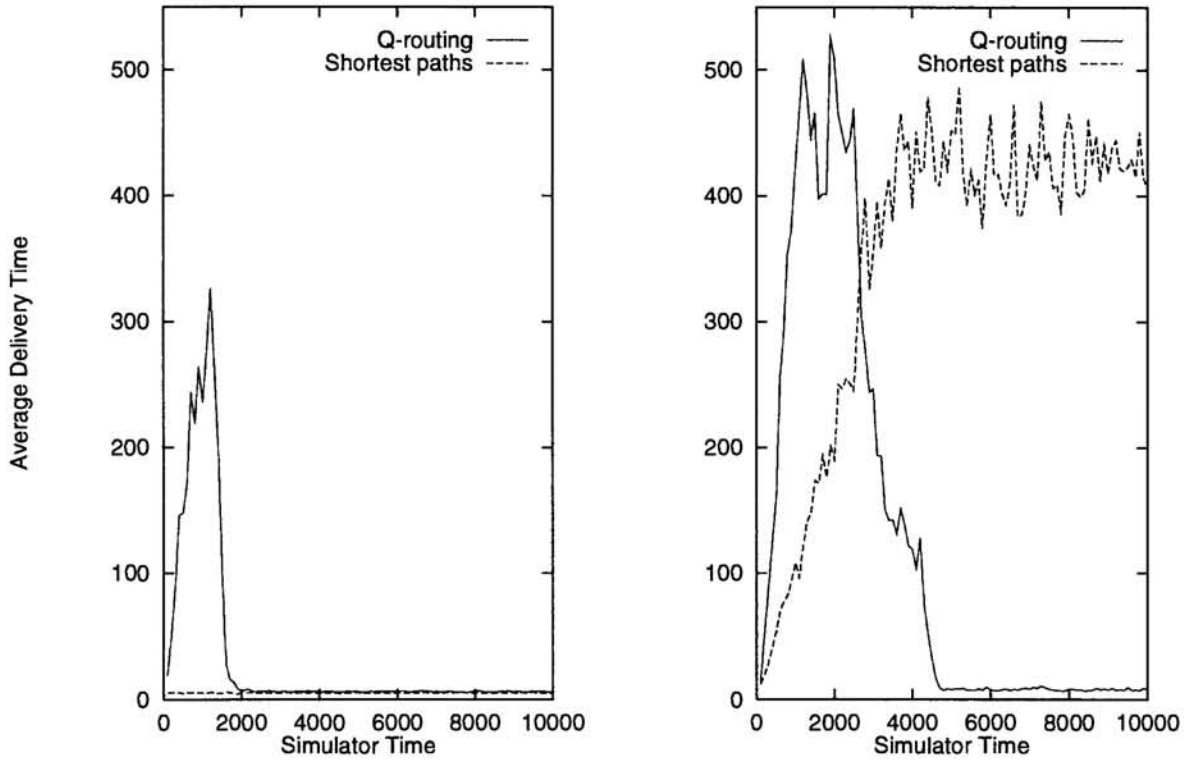

Figure 2: Performance under low load and high load

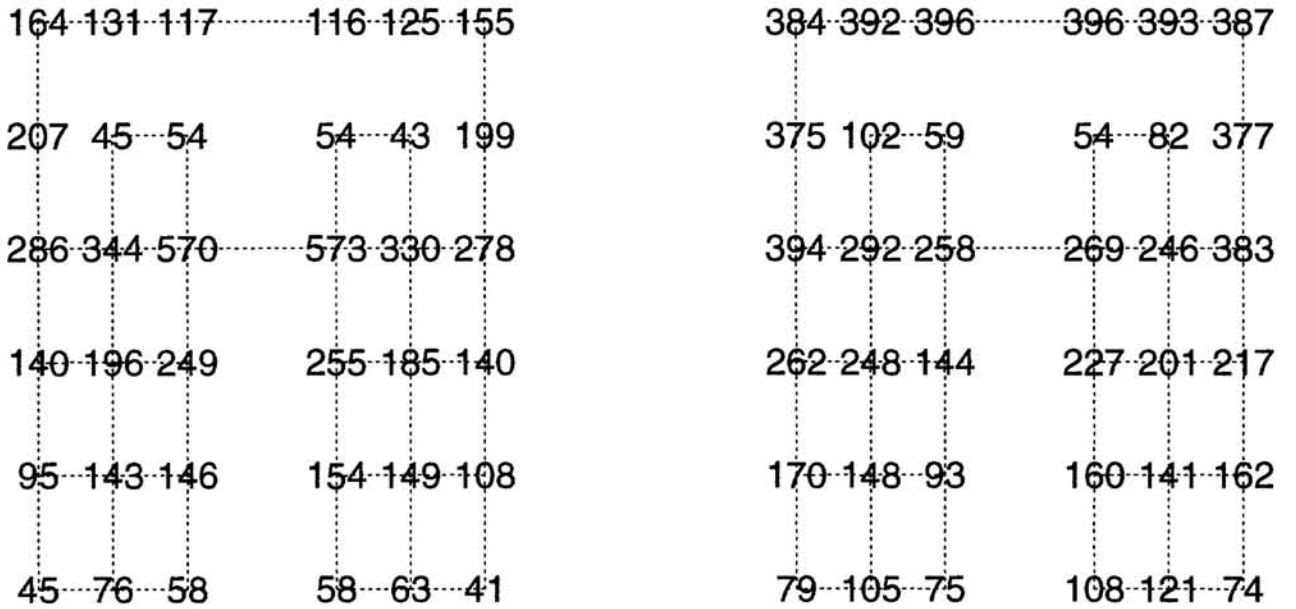

Figure 3: Policy summaries: shortest path and Q-routing under high load

some traffic over a longer than necessary path (across the top of the network) so as to avoid congestion in the center of the network.

The basic result is captured in Figure 4, which compares the performances of the shortest path policy and Q-routing learned policy at various levels of network load. Each point represents the median (over 19 trials) of the mean packet delivery time after learning has settled. When the load is very low, the Q-routing algorithm routes nearly as efficiently as the shortest path policy. As load increases, the shortest path policy leads to exploding levels of network congestion, whereas the learning algorithm continues to route efficiently. Only after a further significant increase in load does the Q-routing algorithm, too, succumb to congestion.

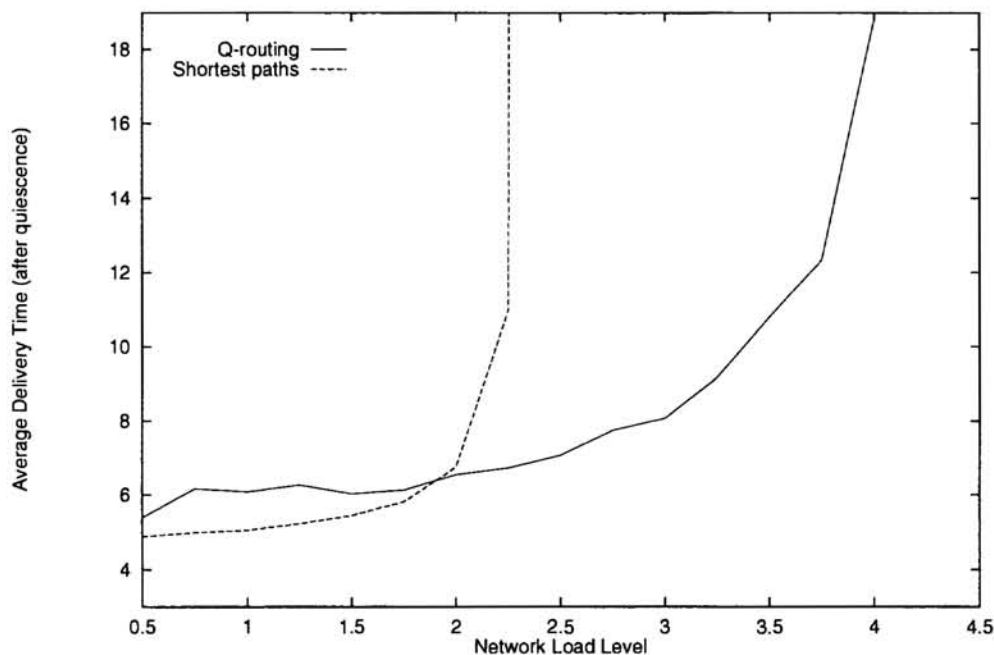

Figure 4: Delivery time at various loads for Q-routing and shortest paths

## 3.1  DYNAMICALLY CHANGING NETWORKS

One advantage a learning algorithm has over a static routing policy is the potential for adapting to changes in crucial system parameters during network operation. We tested the Q-routing algorithm, unmodified, on networks whose topology, traffic patterns, and load level were changing dynamically:

**Topology** We manually disconnected links from the network during simulation. Qualitatively, Q-routing reacted quickly to such changes and was able to continue routing traffic efficiently.

**Traffic patterns** We caused the simulation to oscillate periodically between two very different request patterns in the irregular grid: one in which all traffic was directed between the upper and lower halves of the network, and one in which all traffic was directed between the left and right halves. Again,

after only a brief period of inefficient routing each time the request pattern switched, the Q-routing algorithm adapted successfully.

**Load level** When the overall level of network traffic was raised during simulation, Q-routing quickly adapted its policy to route packets around new bottlenecks. However, when network traffic levels were then lowered again, adaptation was much slower, and never converged on the optimal shortest paths. This effect is discussed in the next section.

## 3.2 EXPLORATION

Given the similarity between the Q-routing update equation and the Bellman-Ford recurrence for shortest paths, it seems surprising that there is any difference whatsoever between the performance of Q-routing and shortest paths routing at low load, as is visible in Figure 4. However, a close look at the algorithm reveals that Q-routing cannot fine-tune a policy to discover shortcuts, since only the best neighbor's estimate is ever updated. For instance, if a node learns an overestimate of the delivery time for an optimal route, then it will select a suboptimal route as long as that route's delivery time is less than the erroneous estimate of the optimal route's delivery time.

This drawback of greedy Q-learning is widely recognized in the reinforcement learning community, and several exploration techniques have been suggested to overcome it [9]. A common one is to have the algorithm select actions with some amount of randomness during the initial learning period[10]. But this approach has two serious drawbacks in the context of distributed routing: (1) the network is continuously changing, thus the initial period of exploration never ends; and more significantly, (2) random traffic has an extremely negative effect on congestion. Packets sent in a suboptimal direction tend to add to queue delays, slowing down all the packets passing through those queues, which adds further to queue delays, etc. Because the nodes make their policy decisions based on only local information, this increased congestion actually changes the problem the learners are trying to solve.

Instead of sending actual packets in a random direction, a node using the "full echo" modification of Q-routing sends requests for information to its immediate neighbors every time it needs to make a decision. Each neighbor returns a single number—using a separate channel so as to not contribute to network congestion in our model—giving that node's current estimate of the total time to the destination. These estimates are used to adjust the $Q_x(d, y)$ values for each neighbor $y$. When shortcuts appear, or if there are inefficiencies in the policy, this information propagates very quickly through the network and the policy adjusts accordingly.

Figure 5 compares the performance of Q-routing and shortest paths routing with "full echo" Q-routing. At low loads the performance of "full echo" Q-routing is indistinguishable from that of the shortest path policy, as all inefficiencies are purged. Under high load conditions, "full echo" Q-routing outperforms shortest paths but the basic Q-routing algorithm does better still. Our analysis indicates that "full echo" Q-routing constantly changes policy under high load, oscillating between using the upper bottleneck and using the central bottleneck for the majority of cross-network traffic. This behavior is unstable and generally leads to worse routing times under high load.

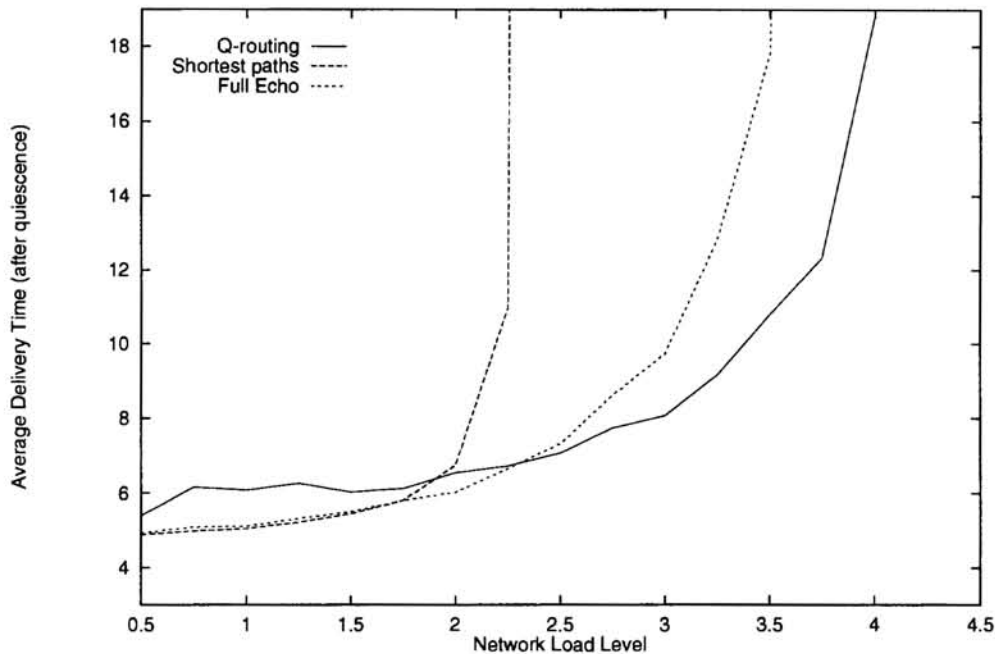

Figure 5: Delivery time at various loads for Q-routing, shortest paths and "full echo" Q-routing

Ironically, the "drawback" of the basic Q-routing algorithm—that it does no exploration and no fine-tuning after initially learning a viable policy—actually leads to improved performance under high load conditions. We still know of no single algorithm which performs best under all load conditions.

# 4    CONCLUSION

This work considers a straightforward application of Q-learning to packet routing. The "Q-routing" algorithm, without having to know in advance the network topology and traffic patterns, and without the need for any centralized routing control system, is able to discover efficient routing policies in a dynamically changing network. Although the simulations described here are not fully realistic from the standpoint of actual telecommunication networks, we believe this paper has shown that adaptive routing is a natural domain for reinforcement learning. Algorithms based on Q-routing but specifically tailored to the packet routing domain will likely perform even better.

One of the most interesting directions for future work is to replace the table-based representation of the routing policy with a function approximator. This could allow the algorithm to integrate more system variables into each routing decision and to generalize over network destinations. Potentially, much less routing information would need to be stored at each node, thereby extending the scale at which the algorithm is useful. We plan to explore some of these issues in the context of packet routing or related applications such as auto traffic control and elevator control.

## Acknowledgements

The authors would like to thank for their support the Bellcore Cognitive Science Research Group, the National Defense Science and Engineering Graduate fellowship program, and National Science Foundation Grant IRI-9214873.

## Footnotes

*Now at Brown University, Department of Computer Science

[1]We denote the function by $Q$ because it corresponds to the $Q$ function used in the reinforcement learning technique of Q-learning [10].

# References

[1] R. Bellman. On a routing problem. *Quarterly of Applied Mathematics*, 16(1):87–90, 1958.

[2] J. Boyan. Modular neural networks for learning context-dependent game strategies. Master's thesis, Computer Speech and Language Processing, Cambridge University, 1992.

[3] L. R. Ford, Jr. *Flows in Networks*. Princeton University Press, 1962.

[4] L.-J. Lin. *Reinforcement Learning for Robots Using Neural Networks*. PhD thesis, School of Computer Science, Carnegie Mellon University, 1993.

[5] M. Littman and J. Boyan. A distributed reinforcement learning scheme for network routing. Technical Report CMU-CS-93-165, School of Computer Science, Carnegie Mellon University, 1993.

[6] H. Rudin. On routing and delta routing: A taxonomy and performance comparison of techniques for packet-switched networks. *IEEE Transactions on Communications*, COM-24(1):43–59, January 1976.

[7] A. Tanenbaum. *Computer Networks*. Prentice-Hall, second edition edition, 1989.

[8] G. Tesauro. Practial issues in temporal difference learning. *Machine Learning*, 8(3/4), May 1992.

[9] Sebastian B. Thrun. The role of exploration in learning control. In David A. White and Donald A. Sofge, editors, *Handbook of Intelligent Control: Neural, Fuzzy, and Adaptive Approaches*. Van Nostrand Reinhold, New York, 1992.

[10] C. Watkins. *Learning from Delayed Rewards*. PhD thesis, King's College, Cambridge, 1989.